# Classifying Patterns of Visual Motion - a Neuromorphic Approach

**Jakob Heinzle** and **Alan Stocker**\*
Institute of Neuroinformatics
University and ETH Zürich
Winterthurerstr. 190, 8057 Zürich, Switzerland
{jakob,alan}@ini.phys.ethz.ch

## Abstract

We report a system that classifies and can learn to classify patterns of
visual motion on-line. The complete system is described by the dynam-
ics of its physical network architectures. The combination of the fol-
lowing properties makes the system novel: Firstly, the front-end of the
system consists of an aVLSI optical flow chip that collectively computes
2-D global visual motion in real-time [1]. Secondly, the complexity of
the classification task is significantly reduced by mapping the continu-
ous motion trajectories to sequences of 'motion events'. And thirdly, all
the network structures are simple and with the exception of the optical
flow chip based on a Winner-Take-All (WTA) architecture. We demon-
strate the application of the proposed generic system for a contactless
man-machine interface that allows to write letters by visual motion. Re-
garding the low complexity of the system, its robustness and the already
existing front-end, a complete aVLSI system-on-chip implementation is
realistic, allowing various applications in mobile electronic devices.

## 1  Introduction

The classification of *continuous* temporal patterns is possible using Hopfield networks with
asymmetric weights [2], but classification is restricted to periodic trajectories with a well-
known start and end point. Also purely feed-forward network architectures were proposed
[3]. However, such networks become unfeasibly large for practical applications.

We simplify the task by first mapping the continuous visual motion patterns to *sequences of
motion events*. A motion event is characterized by the occurrence of visual motion in one
out of a pre-defined set of directions. Known approaches for sequence classification can
be divided into two major categories: The first group typically applies standard Hopfield
networks with time-dependent weight matrices [4, 5]. These networks are relatively ineffi-
cient in storage capacity, using many units per stored pattern. The second approach relies
on time-delay elements and some form of coincidence detectors that respond dominantly
to the correctly time-shifted events of a known sequence [6, 7]. These approaches allow a
compact network architecture. Furthermore, they require neither the knowledge of the start

and end point of a sequence nor a reset of internal states. The sequence classification network of our proposed system is based on the work of Tank and Hopfield [6], but extended to be time-continuous and to show increased robustness. Finally, we modify the network architecture to allow the system to learn arbitrary sequences of a particular length.

## 2  System architecture

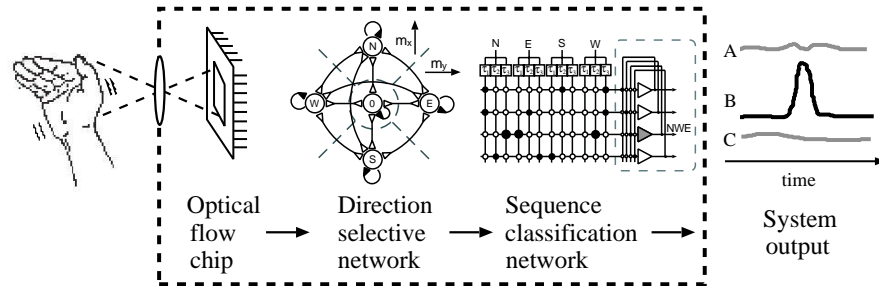

Figure 1: *The complete classification system.*  The input to the system is a real-world moving visual stimulus and the output is the activity of units representing particular trajectory classes.

The system contains three major stages of processing as shown in Figure 1: the optical flow chip estimates global visual motion, the direction selective network (DSN) maps the estimate to motion events and the sequence classification network (SCN) finally classifies the sequences of these events. The architecture reflects the separation of the task into the *classification in motion space* (DSN) and, consequently, the *classification in time* (SCN). Classification in both cases relies on identical WTA networks differing in their inputs only. The outputs of the DSN and the SCN are *'quasi-discrete'* - both signals are continuous-time but due to the non-linear amplification of the WTA represent discrete information.

### 2.1  The optical flow chip

The front-end of the classification system consists of the optical flow chip [1, 8], that estimates 2D visual motion. Due to adaptive circuitry, the estimate of visual motion is fairly independent of illumination conditions. The estimation of visual motion requires the integration of visual information within the image space in order to solve for inherent visual ambiguities. For the purpose of the here presented classification system, the integration of visual information is set to take place over the complete image space. Thus, the resulting estimate represents the *global visual motion* perceived. The output signals of the chip are two analog voltages $m_x$ and $m_y$ that represent *at any instant* the two components of the actual global motion vector. The output signals are linear to the perceived motion within a range of $\pm 0.5$ volts. The resolvable speed range is 1-3500 pix/sec, thus spans more than three orders of magnitude. The continuous-time voltage trajectory $\vec{m}(t) = (m_x(t), m_y(t))$ is the input to the direction selective network.

### 2.2  The direction selective network (DSN)

The second stage transforms the trajectory $\vec{m}(t)$ into a sequence of motion events, where an event means that the motion vector points into a particular region of motion space. Motion space is divided into a set of regions each represented by a unit of the DSN (see Figure 2a). Each direction selective unit (DSU) receives highest input when $\vec{m}$ is within

the corresponding region. In the following we choose four motion directions referred to as north (N), east (E), south (S) and west (W) and a central region for zero motion.

The WTA behavior of the DSN can be described by minimizing the cost function [9]

$$E_{DSN} = \frac{w_{exc}}{2} \sum_i \sum_{j \neq i} V_i V_j + \frac{w_{inh} - w_{exc}}{2} \left( \sum_i V_i - 1 \right)^2 \tag{1}$$

$$- \kappa \sum_i V_i I_i + \frac{1}{R} \sum_i \int_{\frac{1}{2}}^{V_i} g_i^{-1}(V) dV ;$$

where $w_{exc}$ and $w_{inh}$ are the excitatory and inhibitory weights between the DSU [8]. The units have a sigmoidal activation function $V_i = g(v_i)$. Following gradient descent, the dynamics of the units are described by

$$C \frac{dv_i}{dt} = -\frac{1}{R} v_i + w_{exc} V_i - w_{inh} \sum_j V_j + (w_{inh} - w_{exc}) + \kappa I_i, \tag{2}$$

where $C$ and $R$ are the capacitance and resistance of the units. The preferred direction of the $i^{th}$ DSU is given by the angle $\beta_i = \pi(i-1)/2$. The input to the DSU is

$$I_i = \begin{cases} |\vec{m}| \cos^2 \beta_i - \delta & \text{if } |\beta_i - \delta| < \frac{\pi}{2} \\ 0 & \text{if } |\beta_i - \delta| \geq \frac{\pi}{2} \end{cases} \tag{3}$$

where $\vec{m} = (|\vec{m}|, \delta)$ is the motion estimate in polar coordinates. The input to the zero motion unit is $I_0 = I_{thresh} - |\vec{m}|$. In Figure 2b we compare the outputs of a DSU to

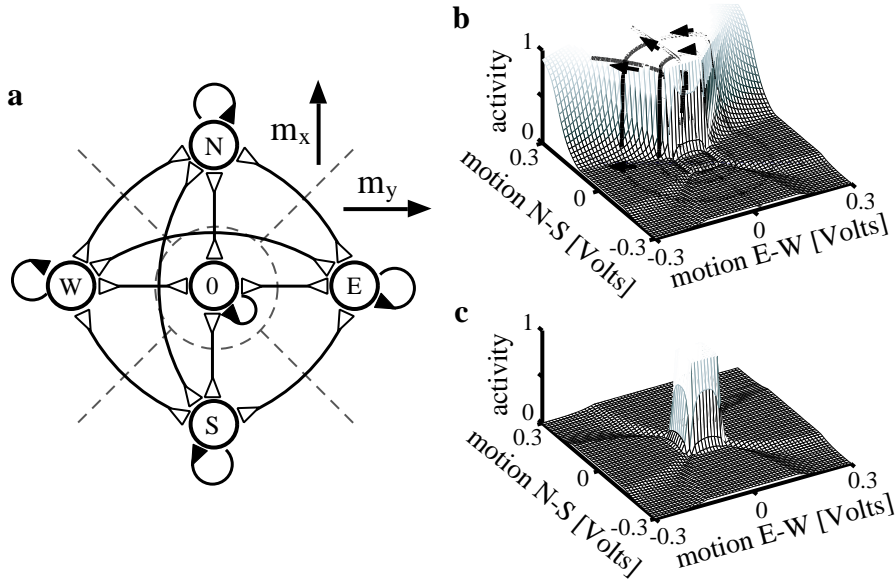

Figure 2: *The direction selective network.* **a)** The WTA architecture of the DSN. Filled connections are excitatory, empty ones are inhibitory. Dotted lines show the regions in motion space where the different units win. **b)** The response of the N-DSU to constant input is shown as surface plot, while the responses of the same unit to dynamic motion trajectories (circles and straight lines) are plotted as lines. Differences between constant and dynamic inputs are marginal. **c)** The output of the zero motion unit to constant input.

constant and varying input $\vec{m}$. The dynamic response is close to the steady state as long as the time-constant of the DSN is smaller than the typical time-scale of $\vec{m}(t)$.

### 2.3 The sequence classification network (SCN)

The classification of the temporal structure of the DSN output is the task of the SCN. The network uses time-delays to "concentrate information in time" [6] (see Figure 3b). In equivalence with the regions in motion space these time-delays form 'regions' in time.

The number of units (SCU) of the SCN is equal to the number of trajectory classes the system is able to classify. We use $k + 1$ time-delays, where $k$ is the number of events of the longest sequence to be classified. The time interval $T_{delay}$ between two maxima of the time-delay functions is the characteristic time-scale of the sequence classification. Again, the SCN is a WTA network with a cost function equivalent to $E_{DSN}$ (1), except that an additional term $\gamma \sum_\mu U_\mu$ is introduced to provide constant input. The SCU have an activation function $U_\mu = g(u_\mu)$ and follow the dynamics

$$C\frac{du_\mu}{dt} = -\frac{1}{R}u_\mu \quad + \quad w_{exc}U_\mu - \sum_\rho w_{inh}U_\rho \tag{4}$$

$$- \quad \gamma + (w_{inh} - w_{exc}) + \sum_i \sum_\nu \Omega_{\mu i;\nu}\widetilde{V}_{i;\nu}.$$

The last term is equivalent to the input term $\kappa I_\mu$ in (2). $\Omega_{\mu i;\nu}$ are the weights of the connections between the DSN and the SCN and $\widetilde{V}_{i;\nu} = \int_0^\infty f_\nu(s)V_i(t-s)ds$ is the delayed output of the DSU. The time-delay functions are the same as in [6][1]. Note that $\gamma$ is the only additional term compared to the dynamics in (2). It allows to set a detection threshold to the sequence classification.

Figure 3a shows an outline of the SCN and its connectivity. For example, if the sequence N-W-E has to be classified, the inputs from the E-DSU delayed by $T_{delay}$, from the W-DSU by $2T_{delay}$ and from the N-DSU by $3T_{delay}$ are excitatory, while all the others are inhibitory. All excitatory as well as all inhibitory weights are equal with excitation being twice as strong as inhibition. The additional time-delay is always inhibitory. It prevents the first motion event from overruling the rest of the sequence and is crucial for the exact classification of short sequences.

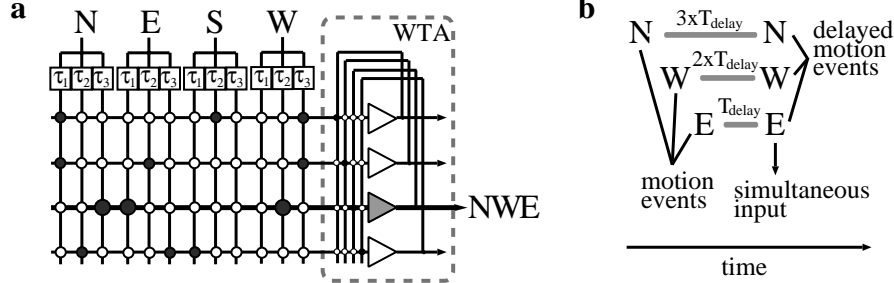

Figure 3: *The sequence classification network.* **a)** Outline of its WTA structure (shown within the dashed line) and its input stage (k=3). The time-delays between the DSU and the SRU are numbered in units of $T_{delay}$. Filled dots are excitatory connections while empty ones are inhibitory. The additional inhibitory delay is not shown. The marked unit recognizes the sequence N-W-E. **b)** A sequence is classified by delaying consecutive motion events such that they provide a simultaneous excitatory input.

# 3   Performance of the system

We measure the performance of the system in two different ways. Firstly, we analyze the robustness to time warping. Knowing the response properties of the optical flow chip [8] we simulate its output to analyze systematically the two other stages of the system. Secondly, we test the complete system including the optical flow chip under real conditions. Here, only a qualitative assessment can be given.

## 3.1   Robustness to time warping

We simulate the visual motion trajectories as a sum of Gaussians in time, thus $\vec{m}(t) = m_0 \sum_j \vec{d_j} \exp -\frac{2(t - j\Delta T)^2}{\delta t}$ where $\vec{d_j} \in [\pm(1, 0), \pm(0, 1)]$. The important parameters are the width of the Gaussians $\delta t$ and the time difference $\Delta T$ between the centers of two neighboring Gaussians. Three schemes are tested: changes of $\delta t$ only, changes of $\Delta T$ only and a linear stretch in time, i.e. a change in both parameters. Time is always measured in units of the characteristic time-delay $T_{delay}$.

For fixed $\Delta T = T_{delay}$, $\delta t$ can be decreased down to $3/5 T_{delay}$ for sequences of length two and down to $1/2 T_{delay}$ for longer sequences. Fixing $\delta t = T_{delay}$, classification is still guaranteed for varying $\Delta T$ according to Figure 4a; *e.g.* for a sequence of length three and input strength $m_0 = 0.2$ volts, $\Delta T$ can maximally increase by 40%. For three and four events (gray and white bars in Figure 4). Linear time stretches change the total input to the system. This causes the asymmetry seen in Figure 4b. Short sequences are relatively more robust to any change in $\Delta T$ than longer sequences[2]

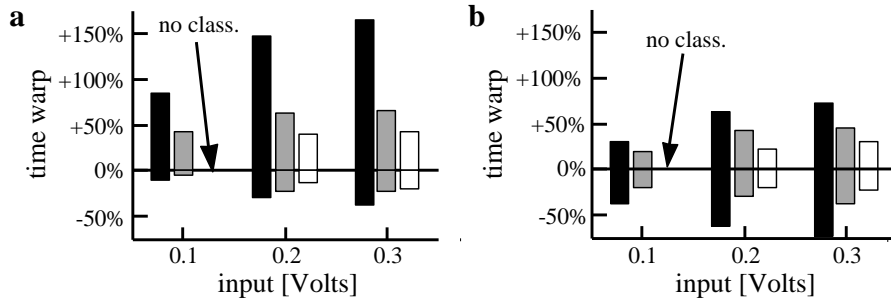

Figure 4: *Time warping.*   The histograms shows the maximal acceptable time warping. The results are shown for three different trajectory lengths (black: two motion events, gray: three events, white: four events) and three different input strengths (maximal output voltages of the optical flow chip). **a)** $\delta t$ is held at $1/2 T_{delay}$ while $\Delta T$ is changed. **b)** Time is stretched linearly and therefore the duration of the events is proportional to $\Delta T$. No classification is possible for sequences of length four at very low input levels.

The system cannot distinguish between the sequences *e.g.* N-W-E-W and N-W-W-W. In this case, the sum of the weighted integrals of the delay functions of both sequences leads to an equivalent input to the SCN. However, if two adjacent events are not allowed to be the same, this problem does not occur.

### 3.2  Real world application - writing letters with patterns of hand movements

The complete system was applied to classify visual motion patterns elicited by hand movements in front of the optical flow chip. Using sequences of three events we are able to classify 36 valid sequences and therefore encode the alphabet. Figure 5 shows a typical visual motion pattern (assigned to the letter 'H') and the corresponding signals at all stages of processing.

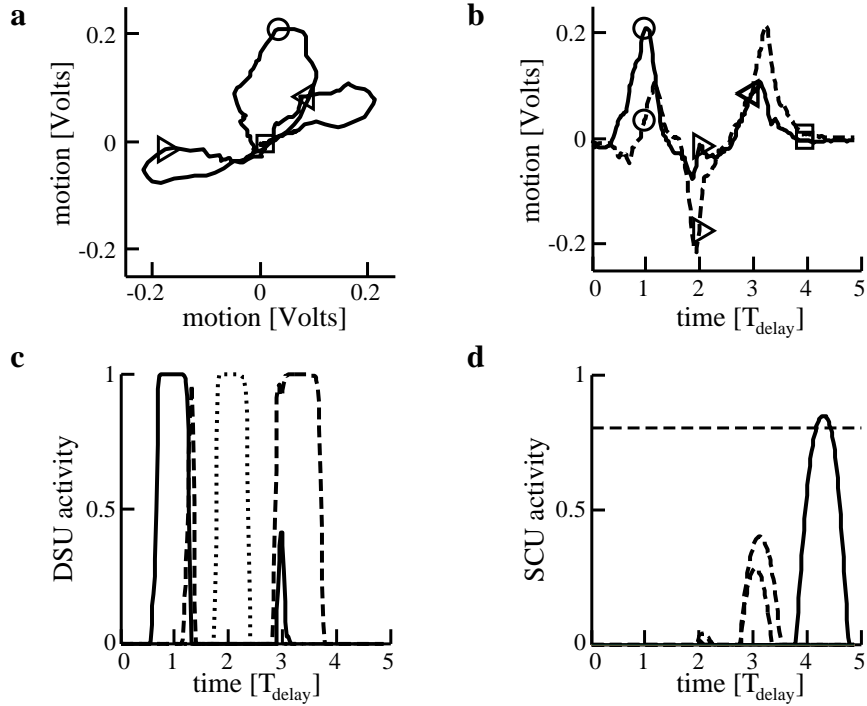

Figure 5: *Tracking a signal through all stages.*  **a)** The output of the optical flow chip to a moving hand in a N-S vs. E-W motion plot. The marks on the trajectory show different time stamps. **b)** The same trajectory including the time stamps in a motion vs. time plot (N-S motion: solid line, E-W motion: dashed line). Time is given in units of $T_{delay}$. **c)** The output of the DSN showing classification in motion space. (N: solid line, E: dashed, W: dotted). **d)** The output of the SCN. Here, the unit that recognizes the trajectory class 'H' is shown by the solid line. The detection threshold is set at 0.8 maximal activity.

The system runs on a 166Mhz Pentium PC using MatLab (TheMathworks Inc.). The signal of the optical flow chip is read into the computer using an AD-card. All simulations are done with simple forward integration of the differential equations.

## 4  Learning motion trajectories

We expanded the system to be able to learn visual motion patterns. We model each set of four synapses connecting the four DSU to a single SCU with the same time-delay by a competitive network of four synapse units (see Figure 6) with very slow time constants. We impose on the output of the four units that their sum $\sum_i \Psi_{\mu i;\nu}$ equals 1. The cost function

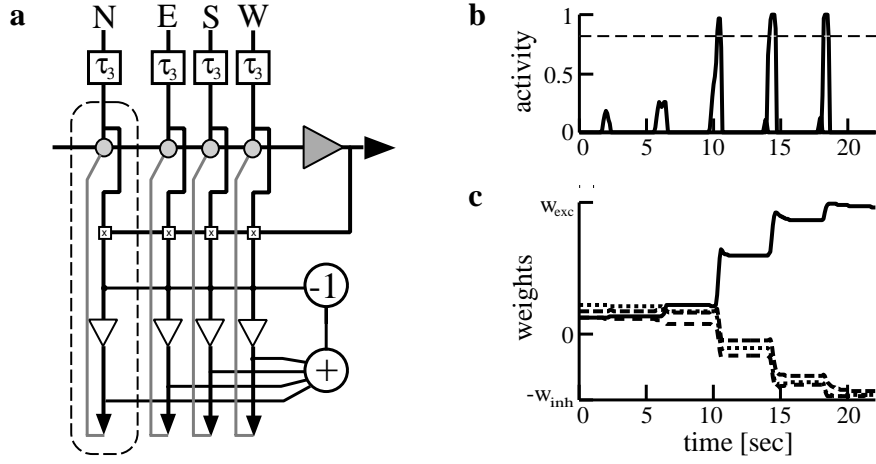

Figure 6: *Learning trajectory classes.* **a)** Schematics of the competitive network of a set of synapses. The dashed line shows one synapse: the synaptic weight $\Omega$, the input to the synapse unit $\psi$ and its output $\Psi$. Multiplication by the output signal of the SCU is indicated by the "x" in the small square, the linear mapping by the bold line from the synapse output to the weight. **b)** Output of the SCU during the repetitive presentation of a particular trajectory. Initial weights were random. **c)** Learning the synaptic weights associated with one particular time-delay.

is given by

$$E_{\mu;\nu} \;=\; \frac{\beta}{2}\Big(\sum_i \Psi_{\mu i;\nu} - 1\Big)^2 + \frac{1}{R}\sum_i \int_{\frac{1}{2}}^{\Psi_{\mu i;\nu}} g^{-1}(\Psi)d\Psi \tag{5}$$
$$\qquad -\; \sum_i \kappa \widetilde{V}_{i;\nu}(1 - \sum_l V_l)U_\mu \Psi_{\mu i;\nu},$$

where the synapse units have an sigmoidal activation function $\Psi = g(\psi)$ and $\widetilde{V}_{i;\nu}$, $V_i$ and $U_\mu$ are defined as in (2) and (4). The synaptic dynamics are given by

$$C\frac{d\psi_{\mu i;\nu}}{dt} = -\frac{1}{R}\psi_{\mu i;\nu} - \beta(\sum_i \Psi_{\mu i;\nu} - 1) + \kappa \widetilde{V}_{i;\nu}(1 - \sum_l V_l)U_\mu. \tag{6}$$

Since the activity of the synapse units $\Psi_{\mu i;\nu}$ is always between 0 and 1 a linear mapping to the actual synaptic weights is performed: $\Omega_{\mu i;\nu} = (w_{exc} + w_{inh})\Psi_{\mu i;\nu} - w_{inh}$. To allow activation of the SCU with unlearned synapses we choose $w_{inh} = (\sum_i \Psi_{\mu i;\nu}^2)w_{inh,ex}$, where $w_{inh,ex}$ is the strongest possible inhibitory weight. This assures that the weights are all slightly positive before learning. $w_{inh}$ increases with increasing learning progress. The input term in (6) is the product of: the input weight ($\kappa$), the delayed input to the synapse ($\widetilde{V}_{i;\nu}$) and the output of the SCU ($U_\mu$) (see Figure 6a). The term $(1 - \sum_i V_i)$ is included to enable learning only if the sequence is completed. The weight of a particular synapse is increased if both, the input to the synapse and the activity of the target SCU are high. The reduction of the other weights is due to the competitive network behavior. The learning mechanism is tested using simulated and real world inputs. Under the restriction that trajectories must differ by more than one event the system is able to learn sequences of length three. Sequences that differ by only one event are learnt by the same SCU, thus subsequent sequences overwrite previous learned ones. In Figure 6b,c the learning process of one particular trajectory class of three events is shown. This trajectory is part of a set of

six trajectories that were learned during one simulation cycle, where each input trajectory was consecutively presented five times.

## 5 Conclusions and outlook

We have shown a strikingly simple[3] network system that reliably classifies distinct visual motion patterns. Clearly, the application of the optical flow chip substantially reduces the remaining computational load and allows real-time processing.

A remarkable feature of our system is that - with the exception of the visual motion front-end, but including the learning rule - all networks have competitive dynamics and are based on the classical Winner-Take-All architecture. WTA networks are shown to be compactly implemented in aVLSI [10]. Thus, given also the small network size, it seems very likely to allow a complete aVLSI *system-on-chip* integration, not considering the learning mechanism. Such a single chip system would represent a very efficient computational device, requiring minimal space, weight and power. The 'quasi-discretization' in visual motion space that emerges from the non-linear amplification in the direction selective network could be refined to include not only more directions but also different speed-levels. That way, richer sets of trajectories can be classified. Many applications in mobile electronic devices are imaginable that require (or desire) a touchless interface. Commercial applications in people control and surveillance seem feasible and are already considered.

### Acknowledgments

This work is supported by the Human Frontiers Science Project grant no. RG00133/2000-B and ETHZ Forschungskredit no. 0-23819-01.

## Footnotes

\*corresponding author; www.ini.unizh.ch/~alan

[1]$f_\nu(t; T_{delay}) = \exp(n)\left(\frac{t}{\nu T_{delay}}\right)^n \exp\left(-n(\frac{t}{\nu T_{delay}})\right)$, where $n = [10..20]$

[2]Imagine the time warp being 20%. For a sequence with five events and more, the time shift becomes larger than $T_{delay}$ for some of the events, which leads to inhibition instead of excitation.

[3]*e.g.* the presented man-machine interface consists only of 31 units and 4x4 time-delays, not counting the network elements in the optical flow chip.

## References

[1] A. Stocker and R. J. Douglas. Computation of smooth optical flow in a feedback connected analog network. *Advances in Neural Information Processing Systems*, 11:706–712, 1999.

[2] L. G. Sotelino, M. Saerens, and H. Bersini. Classification of temporal trajectories by continuous-time recurrent nets. *Neural Networks*, 7(5):767–776, 1994.

[3] D. T. Lin, J. E. Dayhoff, and P. A. Ligomenides. Trajectory recognition with a time-delay neural network. *International Joint Conference on Neural Networks, Baltimore*, III:197–202, 1992.

[4] H. Gutfreund and M. Mezard. Processing of temporal sequences in neural networks. *Phys. Rev. Letters*, 61(2):235–238, July 1988.

[5] D.-L. Lee. Pattern sequence recognition using a time-varying hopfield network. *IEEE Trans. on Neural Networks*, 13(2):330–342, March 2002.

[6] D. W. Tank and J. J. Hopfield. Neural computation by concentrating information in time. *Proc. Natl. Acad. Sci. USA*, 84:1896–1900, April 1987.

[7] J. J. Hopfield and C. D. Brody. What is a moment? Transient synchrony as a collective mechanism for spatiotemporal integration. *Proc. Natl. Acad. Sci. USA*, 98:1282–1287, January 2001.

[8] A. Stocker. *Constraint optimization networks for visual motion perception - analysis and synthesis*. PhD thesis, ETH Zürich, No. 14360, 2001.

[9] J. J. Hopfield. Neurons with graded response have collective computational properties like those of two-state neurons. *Proc. Natl. Acad. Sci. USA*, 81:3088–3092, May 1984.

[10] R. Hahnloser, R. Sarpeshkar, M. Mahowald, R. Douglas, and S. Seung. Digital selection and analogue amplification coexist in a cortex-inspired silicon circuit. *Nature*, 405:947–951, June 2000.

